# Neural Network Model Selection Using Asymptotic Jackknife Estimator and Cross-Validation Method

**Yong Liu**
Department of Physics and
Institute for Brain and Neural Systems
Box 1843, Brown University
Providence, RI, 02912

## Abstract

Two theorems and a lemma are presented about the use of jackknife estimator and the cross-validation method for model selection. Theorem 1 gives the asymptotic form for the jackknife estimator. Combined with the model selection criterion, this asymptotic form can be used to obtain the fit of a model. The model selection criterion we used is the negative of the average predictive likelihood, the choice of which is based on the idea of the cross-validation method. Lemma 1 provides a formula for further exploration of the asymptotics of the model selection criterion. Theorem 2 gives an asymptotic form of the model selection criterion for the regression case, when the parameters optimization criterion has a penalty term. Theorem 2 also proves the asymptotic equivalence of Moody's model selection criterion (Moody, 1992) and the cross-validation method, when the distance measure between response $y$ and regression function takes the form of a squared difference.

## 1 INTRODUCTION

Selecting a model for a specified problem is the key to generalization based on the training data set. In the context of neural network, this corresponds to selecting an architecture. There has been a substantial amount of work in model selection (Lindley, 1968; Mallows, 1973; Akaike, 1973; Stone, 1977; Atkinson, 1978; Schwartz,

1978; Zellner, 1984; MacKay, 1991; Moody, 1992; etc.). In Moody's paper (Moody, 1992), the author generalized Akaike Information Criterion (AIC) (Akaike, 1973) in the regression case and introduced the term *effective number of parameters*. It is thus of great interest to see what the link between this criterion and the cross-validation method (Stone, 1974) is and what we can gain from it, given the fact that AIC is asymptotically equivalent to the cross-validation method (Stone, 1977).

In the method of cross-validation (Stone, 1974), a data set, which has a data point deleted from the original training data set, is used to estimate the parameters of a model by optimizing a parameters optimization criterion. The optimal parameters thus obtained are called the jackknife estimator (Miller, 1974). Then the predictive likelihood of the deleted data point is calculated, based on the estimated parameters. This is repeated for each data point in the original training data set. The fit of the model, or the model selection criterion, is chosen as the negative of the average of these predictive likelihoods. However, the computational cost of estimating parameters for different data point deletion is expensive. In section 2, we obtained an asymptotic formula (theorem 1) for the jackknife estimator based on optimizing a parameters optimization criterion with one data point deleted from the training data set. This somewhat relieves the computational cost mentioned above. This asymptotic formula can be used to obtain the model selection criterion by plugging it into the criterion. Furthermore, in section 3, we obtained the asymptotic form of the model selection criterion for the general case (Lemma 1) and for the special case when the parameters optimization criterion has a penalty term (theorem 2). We also proved the equivalence of Moody's model selection criterion (Moody, 1992) and the cross-validation method (theorem 2). Only sketchy proofs are given when these theorems and lemma are introduced. The detail of the proofs are given in section 4.

## 2   APPROXIMATE JACKKNIFE ESTIMATOR

Let the parameters optimization criterion, with data set $\omega = \{(x_i, y_i),\ i = 1,\ ...,\ n\}$ and parameters $\theta$, be $C_\omega(\theta)$, and let $\omega_{-i}$ denote the data set with $i$th data point deleted from $\omega$. If we denote $\hat{\theta}$ and $\hat{\theta}_{-i}$ as the optimal parameters for criterion $C_\omega(\theta)$ and $C_{\omega_{-i}}(\theta)$, respectively, $\nabla_\theta$ as the derivative with respect to $\theta$ and superscript t as transpose, we have the following theorem about the relationship between $\hat{\theta}$ and $\hat{\theta}_{-i}$.

**Theorem 1** *If the criterion function $C_\omega(\theta)$ is an infinite-order differentiable function and its derivatives are bounded around $\hat{\theta}$. The estimator $\hat{\theta}_{-i}$ (also called jackknife estimator (Miller, 1974)) can be approximated as*

$$\hat{\theta}_{-i} - \hat{\theta} \approx -(\nabla_\theta \nabla_\theta^t C_\omega(\hat{\theta}) - \nabla_\theta \nabla_\theta^t C_i(\hat{\theta}))^{-1} \nabla_\theta C_i(\hat{\theta}) \qquad (1)$$

*in which $C_i(\theta) = C_\omega(\theta) - C_{\omega_{-i}}(\theta)$.*

Proof. Use the Taylor expansion of equation $\nabla_\theta C_{\omega_{-i}}(\hat{\theta}_{-i}) = 0$ around $\hat{\theta}$. Ignore terms higher than the second order.

*Example 1*: Using the *generalized maximum likelihood* method from Bayesian analysis[1] (Berger, 1985), if $\pi(\theta)$ is the prior on the parameters and the observations are mutually independent, for which the distribution is modeled as $y|x \sim f(y|x, \theta)$, the parameters optimization criterion is

$$C_\omega(\theta) = \log[ \prod_{(x_i, y_i) \in \omega} f(y_i|x_i, \theta)\pi(\theta) ] = \sum_{(x_i, y_i) \in \omega} \log f(y_i|x_i, \theta) + \log\pi(\theta). \quad (2)$$

Thus $C_i(\theta) = \log f(y_i|x_i, \theta)$. If we ignore the influence of the deleted data point in the denominator of equation 1, we have

$$\hat{\theta}_{-i} - \hat{\theta} \approx -(\nabla_\theta \nabla_\theta^t C_\omega(\hat{\theta}))^{-1} \nabla_\theta \log f(y_i|x_i, \hat{\theta}). \quad (3)$$

*Example 2*: In the special case of example 1, with noninformative prior $\pi(\theta) = 1$, the criterion is the ordinary log-likelihood function, thus

$$\hat{\theta}_{-i} - \hat{\theta} \approx -[ \sum_{(x_i, y_i) \in \omega} \nabla_\theta \nabla_\theta^t \log f(y_j|x_j, \hat{\theta}) ]^{-1} \nabla_\theta \log f(y_i|x_i, \hat{\theta}). \quad (4)$$

# 3   CROSS-VALIDATION METHOD AND MODEL SELECTION CRITERION

Hereafter we use the negative of the average predictive likelihood, or,

$$\mathcal{T}_m(\omega) = -\frac{1}{n} \sum_{(x_i, y_i) \in \omega} \log f(y_i|x_i, \hat{\theta}_{-i}) \quad (5)$$

as the model selection criterion, in which $n$ is the size of the training data set $\omega$, $m \in \mathcal{M}$ denotes parametric probability models $f(y|x, \theta)$ and $\mathcal{M}$ is the set of all the models in consideration. It is well known that $\mathcal{T}_m(\omega)$ is an unbiased estimator of $r(\theta_0, \hat{\theta}(\cdot))$, the risk of using the model $m$ and estimator $\hat{\theta}$, when the true parameters are $\theta_0$ and the training data set is $\omega$ (Stone, 1974; Efron and Gong, 1983; etc.), i.e.,

$$\begin{aligned} r(\theta_0, \hat{\theta}(\cdot)) &= E\{\mathcal{T}_m(\omega)\} \\ &= E\{-\log f(y|x, \hat{\theta}(\omega))\} \\ &= E\{ -\frac{1}{k} \sum_{(x_j, y_j) \in \omega_n} \log f(y_j|x_j, \hat{\theta}(\omega)) \} \end{aligned} \quad (6)$$

in which $\omega_n = \{(x_j, y_j), j = 1, \dots k\}$ is the test data set, $\hat{\theta}(\cdot)$ is an implicit function of the training data set $\omega$ and it is the estimator we decide to use after we have observed the training data set $\omega$. The expectation above is taken over the randomness of $\omega$, $x$, $y$ and $\omega_n$. The optimal model will be the one that minimizes this criterion. This procedure of using $\hat{\theta}_{-i}$ and $\mathcal{T}_m(\omega)$ to obtain an estimation of risk is often called the cross-validation method (Stone, 1974; Efron and Gong, 1983).

*Remark*: After we have obtained $\hat{\theta}$ for a model, we can use equation 1 to calculate $\hat{\theta}_{-i}$ for each $i$, and put the resulting $\hat{\theta}_{-i}$ into equation 5 to get the fit of the model, thus we will be able to compare different models $m \in \mathcal{M}$.

**Lemma 1** *If the probability model $f(y|x, \theta)$, as a function of $\theta$, is differentiable up to infinite order and its derivatives are bounded around $\hat{\theta}$. The approximation to the model selection criterion, equation 5, can be written as*

$$\mathcal{T}_m(\omega) \approx -\frac{1}{n} \sum_{(x_i, y_i) \in \omega} \log f(y_i | x_i, \hat{\theta}) - \frac{1}{n} \sum_{(x_i, y_i) \in \omega} \nabla_\theta^t \log f(y_i | x_i, \hat{\theta})(\hat{\theta}_{-i} - \hat{\theta}) \qquad (7)$$

Proof. Igoring the terms higher than the second order of the Taylor expansion of $\log f(y_j | x_j, \hat{\theta}_{-i})$ around $\hat{\theta}$ will yield the result.

*Example 2* (continued): Using equation 4, we have, for the model selection criterion,

$$\mathcal{T}_m(\omega) = -\frac{1}{n} \sum_{(x_i, y_i) \in \omega} \log f(y_i | x_i, \hat{\theta}) -$$

$$\frac{1}{n} \sum_{(x_i, y_i) \in \omega} \nabla_\theta^t \log f(y_i | x_i, \hat{\theta}) A^{-1} \nabla_\theta \log f(y_i | x_i, \hat{\theta}). \qquad (8)$$

in which $A = \sum_{(x_j, y_j) \in \omega} \nabla_\theta \nabla_\theta^t \log f(y_j | x_j, \hat{\theta})$. If the model $f(y|x, \theta)$ is the true one, the second term is asymptotically equal to $p$, the number of parameters in the model. So the model selection criterion is

$-$ log-likelihood $+$ number of parameters of the model.

This is the well known Akaike's Information Criterion (AIC) (Akaike, 1973).

*Example 1*(continued): Consider the probability model

$$f(y|x, \theta) = \beta \exp(-\frac{1}{2\sigma^2} \mathcal{E}(y, \eta_\theta(x))) \qquad (9)$$

in which $\beta$ is a normalization factor, $\mathcal{E}(y, \eta_\theta(x))$ is a distance measure between $y$ and regression function $\eta_\theta(x)$. $\mathcal{E}(\cdot)$ as function of $\theta$ is assumed differentiable. Denoting[2] $\mathcal{U}(\theta, \lambda, \omega) = \sum_{(x_i, y_i) \in \omega} \mathcal{E}(y_i, \eta_\theta(x_i)) - 2\sigma^2 \log \pi(\theta | \lambda)$, we have the following theorem,

**Theorem 2** *For the model specified in equation 9 and the parameters optimization criterion specified in equation 2 (example 1), under regular condition, the unbiased estimator of*

$$E\{ \frac{1}{k} \sum_{(x_i, y_i) \in \omega_n} \mathcal{E}(y_i, \eta_{\hat{\theta}}(x_i)) \} \qquad (10)$$

*asymptotically equals to*

$$\frac{1}{n} \sum_{(x_i, y_i) \in \omega} \mathcal{E}(y_i, \eta_{\hat{\theta}}(x_.)) +$$

$$\frac{1}{n} \sum_{(x_i, y_i) \in \omega} \nabla_\theta^t \mathcal{E}(y_i, \eta_{\hat{\theta}}(x_i)) \{ \nabla_\theta \nabla_\theta^t \mathcal{U}(\hat{\theta}, \lambda, \omega) \}^{-1} \nabla_\theta \mathcal{E}(y_i, \eta_{\hat{\theta}}(x_i)). \qquad (11)$$

$$\mathcal{U}(\theta, \lambda, \omega) = \sum_{(x_i, y_i) \in \omega} \mathcal{E}(y_i, \eta_\theta(x_i)) + \lambda \theta^2 + \text{const}(\lambda, \sigma^2).$$

*For the case when $\mathcal{E}(y, \eta_\theta(x)) = (y - \eta_\theta(x))^2$, we get, for the asymptotic equivalency of the equation 11,*

$$\mathcal{E}(\hat{\theta}, \omega) + \frac{2\sigma^2}{n} \frac{1}{2} \times$$

$$\sum_{(x_i, y_i) \in \omega} \frac{\partial}{\partial y_i} \nabla_\theta^t n \mathcal{E}(\hat{\theta}, \omega) \{\nabla_\theta \nabla_\theta^t \mathcal{U}(\hat{\theta}, \lambda, \omega)\}^{-1} \frac{\partial}{\partial y_i} \nabla_\theta n \mathcal{E}(\hat{\theta}, \omega) \qquad (12)$$

*in which $\omega = \{(x_i, y_i), \ i = 1, \ ..., \ n\}$ is the training data set, $\omega_n = \{(x_i, y_i), \ i = 1, \ ..., \ k\}$ is the test data set, and $\mathcal{E}(\theta, \omega) = \frac{1}{n} \sum_{(x_i, y_i) \in \omega} \mathcal{E}(y_i, \eta_\theta(x_i))$.*

Proof. This result comes directly from theorem 1 and lemma 1. Some asymptotic technique has to be used.

*Remark*: The result in equation 12 was first proposed by Moody (Moody, 1992). The *effective number of parameters* formulated in his paper corresponds to the summation in equation 12. Since the result in this theorem comes directly from the asymptotics of the cross-validation method and the jackknife estimator, it gives the equivalency proof between Moody's model selection criterion and the cross-validation method. The detailed proof of this theorem, presented in section 4, is in spirit the same as the one presented in Stone's paper about the proof of the asymptotic equivalence of AIC and the cross-validation method (Stone, 1977).

## 4   DETAILED PROOF OF LEMMAS AND THEOREMS

In order to prove theorem 1, lemma 1 and theorem 2, we will present three auxiliary lemmas first.

**Lemma 2**   *For random variable sequence $x_n$ and $y_n$, if $\lim_{n \to \infty} x_n = x$ and $\lim_{n \to \infty} y_n = x$, then $x_n$ and $y_n$ are asymptotically equivalent.*

Proof. This comes from the definition of asymptotic equivalence. Because asymptotically the two random variable will behave the same as random variable $x$.

**Lemma 3**   *Consider the summation $\sum_i h(x_i, y_i) g(x_i, z)$. If $E(h(x, y)|x, z)$ is a constant $c$ independent of $x$, $y$, $z$, then the summation is asymptotically equivalent to $c \sum_i g(x_i, z)$.*

Proof. According to the theorem of large number,

$$\lim_{n \to \infty} \frac{1}{n} \sum_i h(x_i, y_i) g(x_i, z) = E(h(x, y) g(x, z))$$

$$= E(E(h(x, y)|x, z) g(x, z)) = cE(g(x, z))$$

which is the same as the limit of $\frac{c}{n} \sum_i g(x_i, z)$. Using lemma 2, we get the result of this lemma.

**Lemma 4**   *If $\eta_\theta(\cdot)$ and $g(\theta, \cdot)$ are differentiable up to the second order, and the model $y = \eta_\theta(x) + \epsilon$ with $\epsilon \sim \mathcal{N}(0, \sigma^2)$ is the true model, the second derivative with*

*respect to θ of*

$$\mathcal{U}(\theta, \lambda, \omega) = \sum_{i=1}^{n}(y_i - \eta_\theta(x_i))^2 + g(\theta, \lambda)$$

*evaluated at the minimum of $\mathcal{U}$, i.e., $\hat{\theta}$, is asymptotically independent of random variable $\{y_i, i = 1, ..., n\}$.*

Proof. Explicit calculation of the second derivative of $\mathcal{U}$ with respect to $\theta$, evaluated at $\hat{\theta}$, gives

$$\nabla_\theta \nabla_\theta^t \mathcal{U}(\hat{\theta}, \lambda, \omega) = 2\sum_{i=1}^{n}\nabla_\theta \eta_{\hat{\theta}}(x_i)\nabla_\theta^t \eta_{\hat{\theta}}(x_i) \quad - \quad 2\sum_{i=1}^{n}(y_i - \eta_{\hat{\theta}}(x_i))\nabla_\theta \eta_{\hat{\theta}}(x_i)$$
$$+ \quad \nabla_\theta \nabla_\theta^t g(\hat{\theta}, \lambda)$$

As $n$ approaches infinite, the effect of the second term in $\mathcal{U}$ vanishes, $\hat{\theta}$ approach the mean squared error estimator with infinite amount of data points, or the true parameters $\theta_0$ of the model (consistency of MSE estimator (Jennrich, 1969)), $E(y - \eta_{\hat{\theta}}(x))$ approaches $E(y - \eta_{\theta_0}(x))$ which is 0. According to lemma 2 and lemma 3, the second term of this second derivative vanishes asymptotically. So as $n$ approaches infinite, the second derivative of $\mathcal{U}$ with respect to $\theta$, evaluated at $\hat{\theta}$, approaches

$$\nabla_\theta \nabla_\theta^t \mathcal{U}(\theta_0), \lambda, \omega) = 2\sum_{i=1}^{n}\nabla_\theta \eta_{\theta_0}(x_i)\nabla_\theta^t \eta_{\theta_0}(x_i) + \nabla_\theta \nabla_\theta^t g(\theta_0, \lambda)$$

which is independent of $\{y_i,\ i = 1,\ ...,\ n\}$. According to lemma 2, the result of this lemma is readily obtained.

Now we give the detailed proof of theorem 1, lemma 1 and theorem 2.

**Proof of Theorem 1.** The jackknife estimator $\hat{\theta}_{-i}$ satisfies, $\nabla_\theta C_{\omega_{-i}}(\hat{\theta}_{-i}) = 0$. The Taylor expansion of the left side of this equation around $\hat{\theta}$ gives

$$\nabla_\theta C_{\omega_{-i}}(\hat{\theta}) + \nabla_\theta \nabla_\theta^t C_{\omega_{-i}}(\hat{\theta})(\hat{\theta}_{-i} - \hat{\theta}) + O(|\hat{\theta}_{-i} - \hat{\theta}|^2) = 0$$

According to the definition of $\hat{\theta}$ and $\hat{\theta}_{-i}$, their difference is thus a small quantity. Also because of the boundness of the derivatives, we can ignore higher order terms in the Taylor expansion and get the approximation

$$\hat{\theta}_{-i} - \hat{\theta} \approx -(\nabla_\theta \nabla_\theta^t C_{\omega_{-i}}(\hat{\theta}))^{-1}\nabla_\theta C_{\omega_{-i}}(\hat{\theta})$$

Since $\hat{\theta}$ satisfies $\nabla_\theta C_\omega(\hat{\theta}) = 0$, we can rewrite this equation and obtain equation 1.

**Proof of Lemma 1.** The Taylor expansion of $\log f(y_i|x_i, \hat{\theta}_{-i})$ around $\hat{\theta}$ is

$$\log f(y_i|x_i, \hat{\theta}_{-i}) = \log f(y_i|x_i, \hat{\theta}) + \nabla_\theta^t \log f(y_i|x_i, \hat{\theta})(\hat{\theta}_{-i} - \hat{\theta}) + O(|\hat{\theta}_{-i} - \hat{\theta}|^2)$$

Putting this into equation 5 and ignoring higher order terms for the same argument as that presented in the proof of theorem 1, we readily get equation 7.

**Proof of Theorem 2.** Up to an additive constant dependent only on $\lambda$ and $\sigma^2$, the optimization criterion, or equation 2, can be rewritten as

$$C_\omega(\theta) = -\frac{1}{2\sigma^2}\mathcal{U}(\theta, \lambda, \omega) \tag{13}$$

Now putting equation 9 and 13 into equation 3, we get,

$$\hat{\theta}_{-i} - \hat{\theta} \approx -\{\nabla_\theta \nabla_\theta^t \mathcal{U}(\hat{\theta}, \lambda, \omega)\}^{-1} \nabla_\theta \mathcal{E}(y_i, \eta_{\hat{\theta}}(x_i)) \qquad (14)$$

Putting equation 14 into equation 7, we get, for the model selection criterion,

$$\mathcal{T}_m(\omega) = \frac{1}{n} \sum_{(x_i, y_i) \in \omega} \frac{1}{2\sigma^2} \mathcal{E}(y_i, \eta_{\hat{\theta}}(x_i)) \ +$$

$$\frac{1}{n} \sum_{(x_i, y_i) \in \omega} \frac{1}{2\sigma^2} \nabla_\theta^t \mathcal{E}(y_i, \eta_{\hat{\theta}}(x_i)) \{\nabla_\theta \nabla_\theta^t \mathcal{U}(\hat{\theta}, \lambda, \omega)\}^{-1} \nabla_\theta \mathcal{E}(y_i, \eta_{\hat{\theta}}(x_i)) \quad (15)$$

Recall the discussion associated with equation 6 and now

$$\mathrm{E}\{ -\frac{1}{k} \sum_{(x_j, y_j) \in \omega_n} \log f(y_j | x_j, \hat{\theta}) \} = \mathrm{E}\{ \frac{1}{k} \sum_{(x_j, y_j) \in \omega_n} \frac{1}{2\sigma^2} \mathcal{E}(y_j, \eta_{\hat{\theta}}(x_j)) \} \qquad (16)$$

after some simple algebra, we can obtain the unbiased estimator of equation 10. The result is equation 15 multiplied by $2\sigma^2$, or equation 11. Thus we prove the first part of the theorem.

Now consider the case when

$$\mathcal{E}(y, \eta_\theta(x)) = (y - \eta_\theta(x))^2 \qquad (17)$$

The second term of equation 11 now becomes

$$\frac{1}{n} \sum_{(x_i, y_i) \in \omega} 4(y_i - \eta_{\hat{\theta}}(x_i))^2 \nabla_\theta^t \eta_{\hat{\theta}}(x_i) \{\nabla_\theta \nabla_\theta^t \mathcal{U}(\hat{\theta}, \lambda, \omega)\}^{-1} \nabla_\theta \eta_{\hat{\theta}}(x_i) \qquad (18)$$

As n approaches infinite, $\hat{\theta}$ approach the true parameters $\theta_0$, $\nabla_\theta \eta_{\hat{\theta}}(x.)$ approaches $\nabla_\theta \eta_{\theta_0}(x.)$ and $E((y - \eta_{\hat{\theta}}(x)))^2$ asymptotically equals to $\sigma^2$. Using lemma 4 and lemma 3, we get, for the asymptotic equivalency of equation 18,

$$\frac{1}{n} \sigma^2 \sum_{(x_i, y_i) \in \omega} 2\nabla_\theta^t \eta_{\hat{\theta}}(x_i) \{\nabla_\theta \nabla_\theta^t \mathcal{U}(\hat{\theta}, \lambda, \omega)\}^{-1} 2\nabla_\theta \eta_{\hat{\theta}}(x_i) \qquad (19)$$

If we use notation $\mathcal{E}(\theta, \omega) = \frac{1}{n} \sum_{(x_i, y_i) \in \omega} \mathcal{E}(y_i, \eta_\theta(x_i))$, with $\mathcal{E}(y, \eta_\theta(x))$ of the form specified in equation 17, we can get,

$$\frac{\partial}{\partial y_i} \nabla_\theta n \mathcal{E}(\theta, \omega) = -2\nabla_\theta \eta_\theta(x_i) \qquad (20)$$

Combining this with equation 19 and equation 11, we can readily obtain equation 12.

## 5   SUMMARY

In this paper, we used asymptotics to obtain the jackknife estimator, which can be used to get the fit of a model by plugging it into the model selection criterion. Based on the idea of the cross-validation method, we used the negative of the average predicative likelihood as the model selection criterion. We also obtained the asymptotic form of the model selection criterion and proved that when the parameters optimization criterion is the mean squared error plus a penalty term, this asymptotic form is the same as the form presented by (Moody, 1992). This also served to prove the asymptotic equivalence of this criterion to the method of cross-validation.

## Acknowledgements

The author thanks all the members of the Institute for Brain and Neural Systems, in particular, Professor Leon N Cooper for reading the draft of this paper, and Dr. Nathan Intrator, Michael P. Perrone and Harel Shouval for helpful comments. This research was supported by grants from NSF, ONR and ARO.

## Footnotes

[1]Strictly speaking, it is a method to find the posterior mode.

[2]For example, $\pi(\theta | \lambda) = \mathcal{N}_p(0, \sigma^2/\lambda)$, this corresponds to

## References

Akaike, H. (1973). Information theory and an extension of the maximum likelihood principle. In Petrov and Czaki, editors, *Proceedings of the 2nd International Symposium on Information Theory*, pages 267-281.

Atkinson, A. C. (1978). Posterior probabilities for choosing a regression model. *Biometrika*, 65:39-48.

Berger, J. O. (1985). *Statistical Decision Theory and Bayesian Analysis*. Springer-Verlag.

Efron, B. and Gong, G. (1983). A leisurely look at the bootstrap, the jackknife and cross-validation. *Amer. Stat.*, 37:36-48.

Jennrich, R. (1969). Asymptotic properties of nonlinear least squares estimators. *Ann. Math. Stat.*, 40:633-643.

Lindley, D. V. (1968). The choice of variables in multiple regression (with discussion). *J. Roy. Stat. Soc.*, Ser. B, 30:31-66.

MacKay, D. (1991). *Bayesian methods for adaptive models*. PhD thesis, California Institute of Technology.

Mallows, C. L. (1973). Some comments on Cp. *Technometrics*, 15:661-675.

Miller, R. G. (1974). The jackknife - a review. *Biometrika*, 61:1-15.

Moody, J. E. (1992). The effective number of parameters, an analysis of generalization and regularization in nonlinear learning system. In Moody, J. E., Hanson, S. J., and Lippmann, R. P., editors, *Advances in Neural Information Processing System 4*. Morgan Kaufmann Publication.

Schwartz, G. (1978). Estimating the dimension of a model. *Ann. Stat*, 6:461-464.

Stone, M. (1974). Cross-validatory choice and assessment of statistical predictions (with discussion). *J. Roy. Stat. Soc.*, Ser. B.

Stone, M. (1977). An asymptotic equivalence of choice of model by cross-validation and Akaike's criterion. *J. Roy. Stat. Soc.*, Ser. B, 39(1):44-47.

Zellner, A. (1984). Posterior odds ratios for regression hypotheses: General consideration and some specific results. In Zellner, A., editor, *Basic Issues in Econometrics*, pages 275-305. University of Chicago Press.